# Algebraic Analysis for Non-Regular Learning Machines

**Sumio Watanabe**
Precision and Intelligence Laboratory
Tokyo Institute of Technology
4259 Nagatsuta, Midori-ku, Yokohama 223 Japan
*swatanab@pi.titech.ac.jp*

## Abstract

Hierarchical learning machines are non-regular and non-identifiable statistical models, whose true parameter sets are analytic sets with singularities. Using algebraic analysis, we rigorously prove that the stochastic complexity of a non-identifiable learning machine is asymptotically equal to $\lambda_1 \log n - (m_1 - 1) \log \log n + \text{const.}$, where $n$ is the number of training samples. Moreover we show that the rational number $\lambda_1$ and the integer $m_1$ can be algorithmically calculated using resolution of singularities in algebraic geometry. Also we obtain inequalities $0 < \lambda_1 \le d/2$ and $1 \le m_1 \le d$, where $d$ is the number of parameters.

## 1 Introduction

Hierarchical learning machines such as multi-layer perceptrons, radial basis functions, and normal mixtures are non-regular and non-identifiable learning machines. If the true distribution is almost contained in a learning model, then the set of true parameters is not one point but an analytic variety [4][9][3][10]. This paper establishes the mathematical foundation to analyze such learning machines based on algebraic analysis and algebraic geometry.

Let us consider a learning machine represented by a conditional probability density $p(x|w)$ where $x$ is an $M$ dimensional vector and $w$ is a $d$ dimensional parameter. We assume that $n$ training samples $X^n = \{X_i ; i = 1, 2, ..., n\}$ are independently taken from the true probability distribution $q(x)$, and that the set of true parameters

$$W_0 = \{w \in W ; \ p(x|w) = q(x) \quad (\text{a.s. } q(x)) \ \}$$

is not empty. In Bayes statistics, the estimated distribution $p(x|X^n)$ is defined by

$$p(x|X^n) = \int p(x|w) \, \rho_n(w) dw, \quad \rho_n(w) = \frac{1}{Z_n} \prod_{i=1}^{n} p(X_i|w) \, \varphi(w),$$

where $\varphi(w)$ is an *a priori* probability density on $R^d$, and $Z_n$ is a normalizing constant. The *generalization error* is defined by

$$K(n) = E_{X^n} \{ \int q(x) \, \log \frac{q(x)}{p(x|X^n)} \, dx \}$$

where $E_{X^n}\{\cdot\}$ shows the expectation value over all training samples $X^n$. One of the main purposes in learning theory is to clarify how fast $K(n)$ converges to zero as $n$ tends to infinity. Using the log-loss function $h(x, w) = \log q(x) - \log p(x, w)$, we define the Kullback distance and the empirical one,

$$H(w) = \int h(x, w)q(x)dx, \quad H(w, X^n) = \frac{1}{n}\sum_{i=1}^{n} h(X_i, w).$$

Note that the set of true parameters is equal to the set of zeros of $H(w)$, $W_0 = \{w \in W \; ; \; H(w) = 0\}$. If the true parameter set $W_0$ consists of only one point, the learning machine $p(x|w)$ is called *identifiable*, if otherwise *non-identifiable*. It should be emphasized that, in non-identifiable learning machines, $W_0$ is not a manifold but an *analytic set* with singular points, in general. Let us define the *stochastic complexity* by

$$F(n) = -E_{X^n}\{\log \int \exp(-nH(w, X^n))\varphi(w)dw\}. \tag{1}$$

Then we have an important relation between the stochastic complexity $F(n)$ and the generalization error $K(n)$

$$K(n) = F(n+1) - F(n),$$

which represents that $K(n)$ is equal to the increase of $F(n)$ [1]. In this paper, we show the rigorous asymptotic form of the stochastic complexity $F(n)$ for general non-identifiable learning machines.

## 2 Main Results

We need three assumptions upon which the main results are proven.

(A.1) The probability density $\varphi(w)$ is infinite times continuously differentiable and its support, $W \equiv \text{supp } \varphi$, is compact. In other words, $\varphi \in C_0^\infty$.

(A.2) The log loss function, $h(x, w) = \log q(x) - \log p(x, w)$, is continuous for $x$ in the support $Q \equiv \text{supp} q$, and is analytic for $w$ in an open set $W' \supset W$.

(A.3) Let $\{r_j(x, w^*); j = 1, 2, ..., d\}$ be the associated convergence radii of $h(x, w)$ at $w^*$, in other words, Taylor expansion of $h(x, w)$ at $w^* = (w_1^*, ..., w_d^*)$,

$$h(x, w) = \sum_{k_1,..,k_d=0}^{\infty} a_{k_1 k_2 ... k_d}(x)(w_1 - w_1^*)^{k_1}(w_2 - w_2^*)^{k_2} \cdots (w_d - w_d^*)^{k_d},$$

absolutely converges in $|w_j - w_j^*| < r_j(x, w^*)$. Assume $\inf_{x \in Q} \inf_{w^* \in W} r_j(x, w^*) > 0$ for $j = 1, 2, ..., d$.

**Theorem 1** *Assume (A.1),(A.2), and (A.3). Then, there exist a rational number $\lambda_1 > 0$, a natural number $m_1$, and a constant $C$, such that*

$$|F(n) - \lambda_1 \log n + (m_1 - 1)\log\log n| < C$$

*holds for an arbitrary natural number $n$.*

**Remarks.** (1) If $q(x)$ is compact supported, then the assumption (A.3) is automatically satisfied. (2) Without assumptions (A.1) and (A.3), we can prove the upper bound, $F(n) \le \lambda_1 \log n - (m_1 - 1)\log\log n + const.$

From Theorem 1, if the generalization error $K(n)$ has the asymptotic expansion, then it should be

$$K(n) = \frac{\lambda_1}{n} - \frac{m_1 - 1}{n \log n} + o(\frac{1}{n \log n}).$$

As is well known, if the model is identifiable and has the positive definite Fisher information matrix, then $\lambda_1 = d/2$ ($d$ is the dimension of the parameter space) and $m_1 = 1$. However, hierarchical learning models such as multi-layer perceptrons, radial basis functions, and normal mixtures have smaller $\lambda_1$ and larger $m_1$, in other words, hierarchical models are better learning machines than regular ones if Bayes estimation is applied. Constants $\lambda_1$ and $m_1$ are characterized by the following theorem.

**Theorem 2** *Assume the same conditions as theorem 1. Let $\epsilon > 0$ be a sufficiently small constant. The holomorphic function in Re(z) > 0,*

$$J(z) = \int_{H(w) < \epsilon} H(w)^z \varphi(w) dw,$$

*can be analytically continued to the entire complex plane as a meromorphic function whose poles are on the negative part of the real axis, and the constants $-\lambda_1$ and $m_1$ in theorem 1 are equal to the largest pole of $J(z)$ and its multiplicity, respectively.*

The proofs of above theorems are explained in the following section. Let $w = g(u)$ is an arbitrary analytic function from a set $U \subset R^d$ to $W$. Then $J(z)$ is invariant under the mapping,

$$\{H(w), \varphi(w)\} \rightarrow \{H(g(u)), \varphi(g(u))|g'(u)|\},$$

where $|g'(u)| = |\det(\partial w_i / \partial u_j)|$ is Jacobian. This fact shows that $\lambda_1$ and $m_1$ are invariant under a bi-rational mapping. In section 4, we show an algorithm to calculate $\lambda_1$ and $m_1$ by using this invariance and resolution of singularities.

## 3   Mathematical Structure

In this section, we present an outline of the proof and its mathematical structure.

### 3.1   Upper bound and b-function

For a sufficiently small constant $\epsilon > 0$, we define $F^*(n)$ by

$$F^*(n) = -\log \int_{H(w) < \epsilon} \exp(-nH(w)) \, \varphi(w) \, dw.$$

Then by using the *Jensen's inequality*, we obtain $F(n) \leq F^*(n)$. To evaluate $F^*(n)$, we need the b-function in algebraic analysis [6][7]. Sato, Bernstein, Björk, and Kashiwara proved that, for an arbitrary analytic function $H(w)$, there exist a differential operator $D(w, \partial_w, z)$ which is a polynomial for $z$, and a polynomial $b(z)$ whose zeros are rational numbers on the negative part of the real axis, such that

$$D(w, \partial_w, z)H(w)^{z+1} = b(z)H(w)^z \tag{2}$$

for any $z \in C$ and any $w \in W_\epsilon = \{w \in W; H(w) < \epsilon\}$. By using the relation eq.(2), the holomorphic function $J(z)$ in Re(z) > 0,

$$J(z) \equiv \int_{H(w) < \epsilon} H(w)^z \varphi(w) dw = \frac{1}{b(z)} \int_{H(w) < \epsilon} H(w)^{z+1} D_w^* \varphi(w) dw,$$

can be analytically continued to the entire complex plane as a meromorphic function whose poles are on the negative part of the real axis. The poles, which are rational numbers and ordered from the origin to the minus infinity, are referred to as $-\lambda_1, -\lambda_2, -\lambda_3, ...$, and their multiplicities are also referred to as $m_1, m_2, m_3, ...$ Let $c_{km}$ be the coefficient of the $m$-th order of Laurent expansion of $J(z)$ at $-\lambda_k$. Then,

$$J_K(z) \equiv J(z) - \sum_{k=1}^{K} \sum_{m=1}^{m_k} \frac{c_{km}}{(z+\lambda_k)^{-m}} \tag{3}$$

is holomorphic in $\text{Re}(z) > -\lambda_{K+1}$, and $|J_K(z)| \to 0$ ($|z| \to \infty$, $\text{Re}(z) > -\lambda_{K+1}$). Let us define a function

$$I(t) = \int \delta(t - H(w))\varphi(w)dw$$

for $0 < t < \epsilon$ and $I(t) = 0$ for $\epsilon \leq t \leq 1$. Then $I(t)$ connects the function $F^*(n)$ with $J(z)$ by the relations,

$$J(z) = \int_0^1 t^z\, I(t)\, dt,$$

$$F^*(n) = -\log \int_0^1 \exp(-nt)\, I(t)\, dt.$$

The inverse Laplace transform gives the asymptotic expansion of $I(t)$ as $t \to 0$,

$$I(t) = \sum_{k=1}^{\infty} \sum_{m=1}^{m_k} \frac{c_{km}}{(m-1)!}\, t^{\lambda_k-1}\, (-\log t)^{m-1},$$

resulting in the asymptotic expansion of $F^*(n)$,

$$\begin{aligned} F^*(n) &= -\log \int_0^n \exp(-t)\, I(\frac{t}{n})\, \frac{dt}{n} \\ &= \lambda_1 \log n - (m_1 - 1)\log\log n + O(1), \end{aligned}$$

which is the upper bound of $F(n)$.

## 3.2 Lower Bound

We define a random variable

$$A(X^n) = \sup_{w \in W} \mid n^{1/2}(H(w, X^n) - H(w)) / H(w)^{1/2} \mid. \tag{4}$$

Then, we prove in Appendix that there exists a constant $c_0$ which is independent of $n$ such that

$$E_{X^n}\{A(X^n)^2\} < c_0. \tag{5}$$

By using an inequality $ab \leq (a^2 + b^2)/2$,

$$nH(w, X^n) \geq nH(w) - A(X^n)(nH(w))^{1/2} \geq \frac{1}{2}\{nH(w) - A(X^n)^2\},$$

which derives a lower bound,

$$\begin{aligned} F(n) &\geq -E_{X^n}\{\log \int \exp(-\frac{1}{2}\{nH(w) - A(X^n)^2\})\varphi(w)dw\} \\ &= -\frac{1}{2}E_{X^n}\{A(X^n)^2\} - \log \int \exp(-\frac{nH(w)}{2})\varphi(w)dw \end{aligned} \tag{6}$$

The first term in eq.(6) is bounded. Let the second term be $F_*(n)$, then

$$
\begin{aligned}
F_*(n) &= -\log(Z_1 + Z_2) \\
Z_1 &= \int_{H(w)<\epsilon} \exp(-\frac{nH(w)}{2})\,\varphi(w)dw \cong const.\, n^{-\lambda_1}\,(\log n)^{m_1-1} \\
Z_2 &= \int_{H(w)\geq\epsilon} \exp(-\frac{nH(w)}{2})\,\varphi(w)dw \leq \exp(-\frac{n\epsilon}{2}),
\end{aligned}
$$

which proves the lower bound of $F(n)$,

$$
F(n) \geq \lambda_1 \log n - (m_1 - 1)\log\log n + const.
$$

## 4  Resolution of Singularities

In this section, we construct a method to calculate $\lambda_1$ and $m_1$. First of all, we cover the compact set $W_0$ with a finite union of open sets $W^\alpha$. In other words, $W_0 \subset \cup_\alpha W^\alpha$. Hironaka's resolution of singularities [5][2] ensures that, for an arbitrary analytic function $H(w)$, we can algorithmically find an open set $U^\alpha \subset R^d$ ($U^\alpha$ contains the origin) and an analytic function $g_\alpha : U^\alpha \to W^\alpha$ such that

$$
H(g_\alpha(u)) = a(u)\, u_1^{k_1}\, u_2^{k_2} \cdots u_d^{k_d} \quad (u \in U^\alpha) \tag{7}
$$

where $a(u) > 0$ is a positive function and $k_i \geq 0$ ($1 \leq i \leq d$) are even integers ($a(u)$ and $k_i$ depend on $U^\alpha$). Note that Jacobian $|g'_\alpha(u)| = 0$ if and only if $u \in g_\alpha^{-1}(W_0)$.

$$
\varphi(g_\alpha(u))|g'_\alpha(u)| = \sum_{(p_1,p_2,\ldots,p_d)}^{\text{finite}} c_{p_1,p_2,\ldots,p_d}\, u_1^{p_1} u_2^{p_2} \cdots u_d^{p_d} + R(u), \tag{8}
$$

By combining eq.(7) with eq.(8), we obtain

$$
\begin{aligned}
J_\alpha(z) &\equiv \int_{W^\alpha} H(w)^z \varphi(w) \\
&= \int_{U^\alpha} a(u)\,\{u_1^{k_1}\, u_2^{k_2} \cdots u_d^{k_d}\}^z\, u_1^{p_1} u_2^{p_2} \cdots u_d^{p_d}\, du_1\, du_2 \cdots du_d.
\end{aligned}
$$

For real $z$, $\max_\alpha J_\alpha(z) \leq J(z) \leq \sum_\alpha J_\alpha(z)$,

$$
\lambda_1 = \min_\alpha \min_{(p_1,\ldots,p_d)} \min_{1\leq q\leq d} \frac{p_q+1}{k_q}
$$

and $m_1$ is equal to the number of $q$ which attains the minimum, $\min_{1\leq q\leq d}$.

**Remark.** In a neighborhood of $w_0 \in W_0$, the analytic function $H(w)$ is equivalent to a polynomial $H_{w_0}(w)$, in other words, there exists constants $c_1, c_2 > 0$ such that $c_1 H_{w_0}(w) \leq H(w) \leq c_2 H_{w_0}(w)$. Hironaka's theorem constructs the resolution map $g_\alpha$ for any polynomial $H_{w_0}(w)$ algorithmically in the finite procedures ( blowing-ups for nonsingular manifolds in singularities are recursively applied [5]). From the above discussion, we obtain an inequality, $1 \leq m \leq d$. Moreover there exists $\gamma > 0$ such that $H(w) \leq \gamma|w - w_0|^2$ in the neighborhood of $w_0 \in W_0$, we obtain $\lambda_1 \leq d/2$.

**Example.** Let us consider a model $(x, y) \in R^2$ and $w = (a, b, c, d) \in R^4$,

$$
\begin{aligned}
p(x, y|w) &= p_0(x)\, \frac{1}{(2\pi)^{1/2}} \exp(-\frac{1}{2}(y - \psi(x, w))^2), \\
\psi(x, a, b, c, d) &= a\tanh(bx) + c\tanh(dx),
\end{aligned}
$$

where $p_0(x)$ is a compact support probability density (not estimated). We also assume that the true regression function is $y = \psi(x, 0, 0, 0, 0)$. The set of true parameters is

$$W_0 = \{E_x \psi(X, a, b, c, d)^2 = 0\} = \{ab + cd = 0 \text{ and } ab^3 + cd^3 = 0\}.$$

Assumptions (A.1),(A.2), and (A.3) are satisfied. The singularity in $W_0$ which gives the smallest $\lambda_1$ is the origin and the average loss function in the neighborhood $W^o$ of the origin is equivalent to the polynomial $H_0(a, b, c, d) = (ab + cd)^2 + (ab^3 + cd^3)^2$, (see[9]). Using blowing-ups, we find a map $g : (x, y, z, w) \mapsto (a, b, c, d)$,

$$a = x, \quad b = y^3 w - yzw, \quad c = zwx, \quad d = y,$$

by which the singularity at the origin is resolved.

$$
\begin{aligned}
J(z) &= \int_{W^o} H_0(a, b, c, d)^z \varphi(a, b, c, d) da\, db\, dc\, dd \\
&= \int \{ x^2 y^6 w^2 [1 + (z + w^2(y^2 - z)^3)^2] \}^z |xy^3 w| \varphi(g(x, y, z, w))\, dxdydzdw,
\end{aligned}
$$

which shows that $\lambda_1 = 2/3$ and $m_1 = 1$, resulting that $F(n) = (2/3) \log n + Const$. If the generalization error can be asymptotically expanded, then $K(n) \cong (2/3n)$.

## 5    Conclusion

Mathematical foundation for non-identifiable learning machines is constructed based on algebraic analysis and algebraic geometry. We obtained both the rigorous asymptotic form of the stochastic complexity and an algorithm to calculate it.

## Appendix

In the appendix, we show the inequality eq.(5).

**Lemma 1** *Assume conditions (A.1), (A.2) and (A.3). Then*

$$E_{X^n} \{ \sup_{w \in W} | \frac{1}{\sqrt{n}} \sum_{i=1}^{n} [\, h(X_i, w) - E_x h(X, w)\,]\, |^2 \} < \infty.$$

This lemma is proven by using just the same method as [10]. In order to prove (5), we divide '$\sup_{w \in W}$' in eq.(4) into '$\sup_{H(w) \geq \epsilon}$' and '$\sup_{H(w) < \epsilon}$'. Finiteness of the first half is directly proven by Lemma 1. Let us prove the second half is also finite. We can assume without loss of generality that $w$ is in the neighborhood of $w_0 \in W_0$, because $W$ can be covered by a finite union of neighborhoods. In each neighborhood, by using Taylor expansion of an analytic function, we can find functions $\{f_j(x, w)\}$ and $\{g_j(w) = \prod_i (w_i - w_{0i})^{a_i}\}$ such that

$$h(x, w) = \sum_{j=1}^{J} g_j(w) f_j(x, w), \tag{9}$$

where $\{f_j(x, w_0)\}$ are linearly independent functions of $x$ and $g_j(w_0) = 0$. Since $g_j(w) f_j(x, w)$ is a part of Taylor expansion among $w_0$, $f_j(x, w)$ satisfies

$$E_{X^n} \{ \sup_{w \in W_\epsilon} | \frac{1}{\sqrt{n}} \sum_{i=1}^{n} (f_j(X_i, w) - E_x f_j(X, w))|^2 \} < \infty. \tag{10}$$

By using a definition $\hat{H}(w) \equiv |H(w, X^n) - H(w)|$,

$$
\begin{aligned}
\hat{H}(w)^2 &= |\frac{1}{n} \sum_{i=1}^{n} \{\sum_{j=1}^{J} g_j(w)(f_j(X_i, w) - E_x f_j(X, w))\}|^2 \\
&\leq \sum_{j=1}^{J} g_j(w)^2 \sum_{j=1}^{J} \{\frac{1}{n} \sum_{i=1}^{n} (f_j(X_i, w) - E_x f_j(X, w))\}^2
\end{aligned}
$$

where we used Cauchy-Schwarz's inequality. On the other hand, the inequality $\log x \geq (1/2)(\log x)^2 - x + 1$ $(x > 0)$ shows that

$$
H(w) = \int q(x) \log \frac{q(x)}{p(x, w)} dx \geq \frac{1}{2} \int q(x) (\log \frac{q(x)}{p(x, w)})^2 dx \geq \frac{a_0}{2} \sum_{j=1}^{J} g_j(w)^2
$$

where $a_0 > 0$ is the smallest eigen value of the positive definite symmetric matrix $E_x \{f_j(X, w_0) f_k(X, w_0)\}$. Lastly, combining

$$
A(X^n) = \sup_{w \in W_\epsilon} \frac{n \hat{H}(w)^2}{H(w)} \leq \frac{a_0}{2} \sup_{w \in W_\epsilon} \sum_{j=1}^{J} \{\frac{1}{\sqrt{n}} \sum_{i=1}^{n} (f_j(X_i, w) - E_x f_j(X, w))\}^2
$$

with eq.(10), we obtain eq.(5).

## Acknowledgments

This research was partially supported by the Ministry of Education, Science, Sports and Culture in Japan, Grant-in-Aid for Scientific Research 09680362.

## References

[1] Amari,S., Murata, N.(1993) Statistical theory of learning curves under entropic loss. *Neural Computation*, **5** (4) pp.140-153.

[2] Atiyah, M.F. (1970) Resolution of singularities and division of distributions. *Comm. Pure and Appl. Math.*, **13** pp.145-150.

[3] Fukumizu,K. (1999) Generalization error of linear neural networks in unidentifiable cases.*Lecture Notes in Computer Science*, **1720** Springer, pp.51-62.

[4] Hagiwara,K., Toda,N., Usui,S. (1993) On the problem of applying AIC to determine the structure of a layered feed-forward neural network. *Proc. of IJCNN*, **3** pp.2263-2266.

[5] Hironaka, H. (1964) Resolution of singularities of an algebraic variety over a field of characteristic zero, I,II. *Annals of Math.*, **79** pp.109-326.

[6] Kashiwara, M. (1976) B-functions and holonomic systems, *Invent. Math.*, **38** pp.33-53.

[7] Oaku, T. (1997) An algorithm of computing b-functions. *Duke Math. J.*, **87** pp.115-132.

[8] Sato, M., Shintani,T. (1974) On zeta functions associated with prehomogeneous vector space.*Annals of Math.*, **100**, pp.131-170.

[9] Watanabe, S.(1998) On the generalization error by a layered statistical model with Bayesian estimation. *IEICE Trans.*, **J81-A** pp.1442-1452. English version: *Elect. Comm. in Japan.*, to appear.

[10] Watanabe, S. (1999) Algebraic analysis for singular statistical estimation. *Lecture Notes in Computer Science*, **1720** Springer, pp.39-50.